# Interpreting Neural Response Variability as Monte Carlo Sampling of the Posterior

**Patrik O. Hoyer**∗ **and Aapo Hyvärinen**

Neural Networks Research Centre
Helsinki University of Technology
P.O. Box 9800, FIN-02015 HUT, Finland
http://www.cis.hut.fi/phoyer/
patrik.hoyer@hut.fi

## Abstract

The responses of cortical sensory neurons are notoriously variable, with the number of spikes evoked by identical stimuli varying significantly from trial to trial. This variability is most often interpreted as 'noise', purely detrimental to the sensory system. In this paper, we propose an alternative view in which the variability is related to the uncertainty, about world parameters, which is inherent in the sensory stimulus. Specifically, the responses of a population of neurons are interpreted as stochastic samples from the posterior distribution in a latent variable model. In addition to giving theoretical arguments supporting such a representational scheme, we provide simulations suggesting how some aspects of response variability might be understood in this framework.

## 1 Introduction

During the past half century, a wealth of data has been collected on the response properties of cortical sensory neurons. The majority of this research has focused on how the mean firing rates of individual neurons depend on the sensory stimulus. Similarly, mathematical models have mainly focused on describing how the mean firing rate could be computed from the input. One aspect which this research does not address is the high variability of cortical neural responses. The trial-to-trial variation in responses to identical stimuli are significant [1, 2], and several trials are typically required to get an adequate estimate of the mean firing rate.

The standard interpretation is that this variability reflects 'noise' which limits the accuracy of the sensory system [2, 3]. In the standard model, the firing rate is given by

$$\text{rate} = f(\text{stimulus}) + \text{noise}, \tag{1}$$

where $f$ is the 'tuning function' of the cell in question. Here, the magnitude of the noise may depend on the stimulus. Experimental results [1, 2] seem to suggest that the amount of variability depends only on the mean firing rate, i.e. $f(\text{stimulus})$, and not on the particular

---

∗Current address: 4 Washington Place, Rm 809, New York, NY 10003, USA

stimulus that evoked it. Specifically, spike count variances tend to grow in proportion to spike count means [1, 2]. This has been taken as evidence for something like a Poisson process for neural firing.

This standard view is not completely satisfactory. First, the exquisite sensitivity and the reliability of many peripheral neurons (see, e.g. [3]) show that neurons in themselves need not be very unreliable. In vitro experiments [4] also suggest that the large variability does not have its origin in the neurons themselves, but is a property of intact cortical circuits. One is thus tempted to point at synaptic 'background' activity as the culprit, attributing the variability of individual neurons to variable inputs. This seems reasonable, but it is not quite clear why such modulation of firing should be considered meaningless noise rather than reflecting complex neural computations.

Second, the above model does a poor job of explaining neural responses in the phenomenon known as 'visual competition': When viewing ambiguous (bistable) figures, perception, and the responses of many neurons with it, oscillates between two distinct states (for a review, see [5]). In other words, a single stimulus can yield two very different firing rates in a single neuron depending on how the stimulus is interpreted. In the above model, this means that either (a) the noise term needs to have a bimodal distribution, or (b) we are forced to accept the fact that neurons can be tuned to stimulus *interpretations*, rather than stimuli themselves. The former solution is clearly unattractive. The latter seems sensible, but we have then simply transformed the problem of oscillating firing rates into a problem of oscillating interpretations: Why should there be variability (over time, and over trials) in the interpretation of a stimulus?

What would be highly desirable is a theoretical framework in which the variability of responses could be shown to have a specific purpose. One suggestion [6] is that variability could improve the signal to noise ratio through a phenomenon known as 'stochastic resonance'. Another recent suggestion is that variability contributes to the contrast invariance of visual neurons [7].

In this paper, we will propose an alternative explanation for the variability of neural responses. This hypothesis attempts to account for both aspects of variability described above: the Poisson-like 'noise' and the oscillatory responses to ambiguous stimuli. Our suggestion is based on the idea that cortical circuits implement Bayesian inference in latent variable models [8, 9, 10]. Specifically, we propose that neural firing rates might be viewed as representing Monte Carlo samples from the posterior distribution over the latent variables, given the observed input. In this view, the response variability is related to the uncertainty, about world parameters, which is inherent in any stimulus. This representation would allow not only the coding of parameter values but also of their uncertainties. The latter could be accomplished by pooling responses over time, or over a population of redundant cells.

Our proposal has a direct connection to Monte Carlo methods widely used in engineering. These methods use built-in randomness to solve difficult problems that cannot be solved analytically. In particular, such methods are one of the main options for performing approximate inference in Bayesian networks [11]. With that in mind, it is perhaps even a bit surprising that Monte Carlo sampling has not, to our knowledge, previously been suggested as an explanation for the randomness of neural responses.

Although the approach proposed is not specific to sensory modality, we will here, for concreteness, exclusively concentrate on vision. We shall start by, in the next section, reviewing the basic probabilistic approach to vision. Then we will move on to further explain the proposal of this contribution.

## 2   The latent variable approach to vision

### 2.1   Bayesian models of high-level vision

Recently, a growing number of researchers have argued for a probabilistic approach to vision, in which the functioning of the visual system is interpreted as performing Bayesian inference in latent variable models, see e.g. [8, 9, 10]. The basic idea is that the visual input is seen as the observed data in a probabilistic generative model. The goal of vision is to estimate the latent (i.e. unobserved or hidden) variables that caused the given sensory stimulation.

In this framework, there are a number of world parameters that contribute to the observed data. These could be, for example, object identities, dimensions and locations, surface properties, lighting direction, and so forth. These parameters are not directly available to the sensory system, but must be estimated from the effects that they have on the images projected onto the retinas. Collecting all the unknown world variables into the vector $\mathbf{s}$ and all sensory data into the vector $\mathbf{x}$, the probability that a given set of world parameters caused a given sensory stimulus is

$$p(\mathbf{s}|\mathbf{x}) \propto p(\mathbf{x}|\mathbf{s})p(\mathbf{s}), \tag{2}$$

where $p(\mathbf{s})$ is the *prior probability* of the set of world parameters $\mathbf{s}$, and $p(\mathbf{x}|\mathbf{s})$ describes how sensory data is generated from the world parameters. The distribution $p(\mathbf{s}|\mathbf{x})$ is known as the *posterior distribution*.

A specific perceptual task then consists of estimating some subset of the world variables, given the observed data [10]. In face recognition, for example, one wants to know the identity of a person but one does not care about the specific viewpoint or the direction of lighting. Note, however, that sometimes one might specifically want to estimate viewpoint or lighting, disregarding identity, so one cannot just automatically throw out that information [10]. In a latent variable model, all relevant information is contained in the complete posterior distribution $p(identity, viewpoint, lighting|sensory\ data)$. To estimate the identity one must use the marginal posterior $p(identity|sensory\ data)$, obtained by integrating out the *viewpoint* and *lighting* variables. Bayesian models of high-level vision model the visual system as performing these types of computations, but typically do not specify how they might be neurally implemented.

### 2.2   Neural network models of low-level vision

This probabilistic approach has not only been suggested as an abstract framework for vision, but in fact also as a model for interpreting actual neural firing patterns in the early visual cortex [12, 13]. In this line of research, the hypothesis is that the activity of individual neurons can be associated with hidden state variables, and that the neural circuitry implements probabilistic inference.[1]

The model of Olshausen and Field [12], known as *sparse coding* or *independent component analysis* (ICA) [14], depending on the viewpoint taken, is perhaps the most influential latent variable model of early visual processing to date. The hidden variables $s_j$ are *independent* and *sparse*, such as is given, for instance, by the double-sided exponential distribution $p(s_j) = \exp(-\sqrt{2}|s_j|)/\sqrt{2}$. The observed data vector $\mathbf{x}$ is then given by a linear combination of the $s_j$, plus additive isotropic Gaussian noise. That is, $\mathbf{x} = \mathbf{As} + \mathbf{n}$,

where $\mathbf{A}$ is a matrix of model parameters (weights), and $\mathbf{n}$ is Gaussian with zero mean and covariance matrix $\sigma^2\mathbf{I}$.

How does this abstract probabilistic model relate to neural processing? Olshausen and Field showed that when the model parameters are estimated (learned) from natural image data, the basis vectors (columns of $\mathbf{A}$) come to resemble V1 simple cell receptive fields. Moreover, the latent variables $s_j$ relate to the activities of the corresponding cells. Specifically, Olshausen and Field suggested [12] that the firing rates of the neurons correspond to the *maximum a posteriori* (MAP) estimate of the latent variables, given the image input: $\hat{\mathbf{s}} = \arg\max_{\mathbf{s}} p(\mathbf{s}|\mathbf{x})$.

An important problem with this kind of a MAP representation is that it attempts to represent a complex posterior distribution using only a single point (at the maximum). Such a representation cannot adequately represent multimodal posterior distributions, nor does it provide any way of coding the uncertainty of the value (the width of the peak). Many other proposed neural representations of probabilities face similar problems [11] (however, see [15] for a recent interesting approach to representing distributions). Indeed, it has been said [10, 16] that how probabilities actually are represented in the brain is one of the most important unanswered questions in the probabilistic approach to perception. In the next section we suggest an answer based on the idea that probability distributions might be represented using response variability.

## 3  Neural responses as samples from the posterior distribution?

As discussed in the previous section, the distribution of primary interest to a sensory system is the posterior distribution over world parameters. In all but absolutely trivial models, computing and representing such a distribution requires approximative methods, of which one major option is Monte Carlo methods. These generate stochastic samples from a given distribution, without explicitly calculating it, and such samples can then be used to approximately represent or perform computations on that distribution [11].

Could the brain use a Monte Carlo approach to perform Bayesian inference? If neural firing rates are used (even indirectly) to represent continuous-valued latent variables, one possibility would be for firing rate *variability* to represent a probability distribution over these variables. Here, there are two main possibilities:

(a) *Variability over time*. A single neuron could represent a continuous distribution if its firing rate fluctuated over time in accordance with the distribution to be represented. At each instant in time, the instantaneous firing rate would be a random sample from the distribution to be represented.

(b) *Variability over neurons*. A distribution could be instantaneously represented if the firing rate of each neuron in a pool of identical cells was independently and randomly drawn from the distribution to be represented.

Note that these are not exclusive, both types of variability could potentially coexist. Also note that both cases lead to trial-to-trial variability, as all samples are assumed independent.

Both possibilities have their advantages. The first option is much more efficient in terms of the number of cells required, which is particularly important for representing high-dimensional distributions. In this case, dependencies between variables can naturally be represented as temporal correlations between neurons representing different parameters. This is not nearly as straightforward for case (b). On the other hand, in terms of processing speed, this latter option is clearly preferred to the former. Any decisions should optimally be based on the whole posterior distribution, and in case (a) this would require collecting samples over an extended period of time.

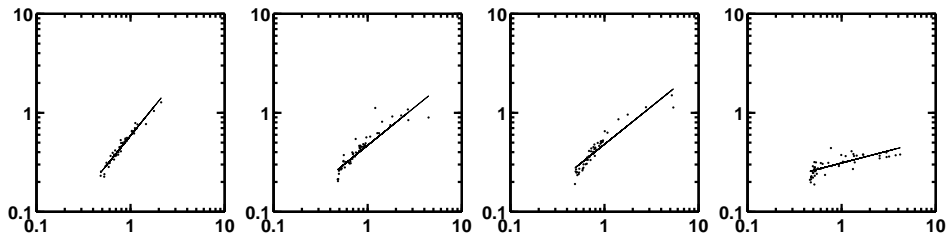

Figure 1: Variance of response versus mean response, on log-log axes, for 4 representative model neurons. Each dot gives the mean (horizontal axis) and variance (vertical axis) of the response of the model neuron in question to one particular stimulus. Note that the scale of responses is completely arbitrary.

We will now explain how both aspects of response variability described in the introduction can be understood in this framework. First, we will show how a simple mean-variance relationship can arise through sampling in the independent component analysis model. Then, we will consider how the variability associated with the phenomenon of visual competion can be interpreted using sampling.

## 3.1 Example 1: Posterior sampling in ICA

Here, we sample the posterior distribution in the ICA model of natural images, and show how this might relate to the conspicious variance-mean relation of neural response variability. First, we used standard ICA methods [17] to estimate a complete basis $\mathbf{A}$ for the 40-dimensional principal subspace of $12 \times 12$ -pixel natural image patches. Motivated by the non-negativity of neural firing rates we modified the model to assume single-sided exponential priors $p(s_j) = \exp(-s_j)$ [18], and augmented the basis so that a pair of neurons coded separately for the positive and negative parts of each original independent component. We then took 50 random natural image patches and sampled the posterior distributions $p(\mathbf{s}|\mathbf{x})$ for all 50 patches $\mathbf{x}$, taking a total of 1000 samples in each case.[2]

From the 1000 collected samples, we calculated the mean and variance of the response of each neuron to each stimulus separately. We then plotted the variance against the mean independently for each neuron in log-log coordinates. Figure 1 shows the plots from 4 randomly selected neurons. The crucial thing to note is that, as for real neurons [1], *the variance of the response is systematically related to the mean response, and does not seem to depend on the particular stimulus used to elicit a given mean response*. This feature of neural variability is perhaps the single most important reason to believe that the variability is meaningless noise inherent in neural firing; yet we have shown that something like this might arise through sampling in a simple probabilistic model.

Following [1, 2], we fitted lines to the plots, modeling the variance as var $= \alpha \cdot \text{mean}^{\beta}$. Over the whole population (80 model neurons), the mean values of $\alpha$ and $\beta$ were 0.49 and 0.85, with population standard deviations 0.07 and 0.23 (respectively). Although these values do not actually match those obtained from physiology (most reports give values of $\alpha$ between 1 and 2, and $\beta$ close to 1, see [1, 2]), this is to be expected. First, the values of these parameters probably depend on the specifics of the ICA model, such as its dimensionality and the noise level; we did not optimize these to attempt to fit physiology. Second, and more importantly, we do not believe that ICA is an exact model of V1 function. Rather, the visual cortex would be expected to employ a much more complicated, hierarchical, image

model. Thus, our main goal was not to show that the particular parameters of the variance-mean relation could be explained in this framework, but rather the surprising fact that such a simple relation might arise as a result of posterior sampling in a latent variable model.

### 3.2   Example 2: Visual competition as sampling

As described in the introduction, in addition to the mean-variance relationship observed throughout the visual cortex, a second sort of variability is that observed in visual competition. This phenomenon arises when viewing a bistable figure, such as the famous Necker cube or Rubin's vase/face figure. These figures each have two interpretations (explanations) that both cannot reasonably explain the image simultaneously. In a latent variable image model, this corresponds to the case of a bimodal posterior distribution.

When such figures are viewed, the perception oscillates between the two interpretations (for a review of this phenomenon, see [5]). This corresponds to jumping from mode to mode in the posterior distribution. This can directly be interpreted as sampling of the posterior. When the stimulus is modified so that one interpretations is slightly more natural than the other one, the former is dominant for a relatively longer period compared with the latter (again, see [5]), just as proper sampling takes relatively more samples from the mode which has larger probability mass. Although the above might be considered purely 'perceptual' sampling, animal studies indicate that especially in higher-level visual areas many neurons modulate their responses in sync with the animal's perceptions [5, 19]. This link proves that some form of sampling is clearly taking place on the level of neural firing rates as well.

Note that this phenomenon might be considered as evidence for sampling scheme (a) and against (b). If we instantaneously could represent whole distributions, we should be able to keep both interpretations in mind simultaneously. This is in fact (weak) evidence against any scheme of representing whole distributions instantaneously, by the same logic.

## 4   Conclusions

One of the key unanswered questions in theoretical neuroscience seems to be: How are probabilities represented by the brain? In this paper, we have proposed that probability distributions might be represented using response variability. If true, this would also present a functional explanation for the significant amount of cortical neural 'noise' observed. Although it is clear that the variability degrades performance on many perceptual tasks of the laboratory, it might well be that it plays an important function in everyday sensory tasks. Our proposal would be one possible way in which it might do so.

Do actual neurons employ such a computational scheme? Although our arguments and simulations suggest that it might be possible (and should be kept in mind), future research will be needed to answer that question. As we see it, key experiments would compare measured firing rate variability statistics (single unit variances, or perhaps two-unit covariances) to those predicted by latent variable models. Of particular interest are cases where contextual information reduces the uncertainty inherent in a given stimulus; our hypothesis predicts that in such cases neural variability is also reduced.

A final question concerns how neurons might actually implement Monte Carlo sampling in practice. Because neurons cannot have global access to the activity of all other neurons in the population, the only possibility seems to be something akin to Gibbs sampling [20]. Such a scheme might require only relatively local information and could thus conceivably be implemented in actual neural networks.

**Acknowledgements** — Thanks to Paul Hoyer, Jarmo Hurri, Bruno Olshausen, Liam Paninski, Phil Sallee, Eero Simoncelli, and Harri Valpola for discussions and comments.

## Footnotes

[1]Here, it must be stressed that in these low-level neural network models, the hidden variables that the neurons represent are not what we would typically consider to be the 'causal' variables of a visual scene. Rather, they are low-level visual features similar to the optimal stimuli of neurons in the early visual cortex. The belief is that more complex hierarchical models will eventually change this.

[2]This was accomplished using a Markov Chain Monte Carlo method, as described in the Appendix. However, the technical details of this method are not very relevant to this argument.

# References

[1] A. F. Dean. The variability of discharge of simple cells in the cat striate cortex. *Experimental Brain Research*, 44:437–440, 1981.

[2] D. J. Tolhurst, J. A. Movshon, and A. F. Dean. The statistical reliability of signals in single neurons in cat and monkey visual cortex. *Vision Research*, 23:775–785, 1983.

[3] A. J. Parker and W. T. Newsome. Sense and the single neuron: Probing the physiology of perception. *Annual Review of Neuroscience*, 21:227–277, 1998.

[4] G. R. Holt, W. R. Softky, C. Koch, and R. J. Douglas. Comparison of discharge variability in vitro and in vivo in cat visual cortex neurons. *Journal of Neurophysiology*, 75:1806–1814, 1996.

[5] R. Blake and N. K. Logothetis. Visual competition. *Nature Reviews Neuroscience*, 3:13–21, 2002.

[6] M. Rudolph and A. Destexhe. Do neocortical pyramidal neurons display stochastic resonance? *Journal of Computational Neuroscience*, 11:19–42, 2001.

[7] J. S. Anderson, I. Lampl, D. C. Gillespie, and D. Ferster. The contribution of noise to contrast invariance of orientation tuning in cat visual cortex. *Science*, 290:1968–1972, 2000.

[8] D. C. Knill and W. Richards, editors. *Perception as Bayesian Inference*. Cambridge University Press, 1996.

[9] R. P. N. Rao, B. A. Olshausen, and M. S. Lewicki, editors. *Probabilistic Models of the Brain*. MIT Press, 2002.

[10] D. Kersten and P. Schrater. Pattern inference theory: A probabilistic approach to vision. In R. Mausfeld and D. Heyer, editors, *Perception and the Physical World*. Wiley & Sons, 2002.

[11] P. Dayan. Recognition in hierarchical models. In F. Cucker and M. Shub, editors, *Foundations of Computational Mathematics*. Springer, Berlin, Germany, 1997.

[12] B. A. Olshausen and D. J. Field. Sparse coding with an overcomplete basis set: A strategy employed by V1? *Vision Research*, 37:3311–3325, 1997.

[13] R. P. N. Rao and D. H. Ballard. Predictive coding in the visual cortex: a functional interpretation of some extra-classical receptive field effects. *Nature Neuroscience*, 2(1):79–87, 1999.

[14] A. J. Bell and T. J. Sejnowski. The 'independent components' of natural scenes are edge filters. *Vision Research*, 37:3327–3338, 1997.

[15] R. S. Zemel, P. Dayan, and A. Pouget. Probabilistic interpretation of population codes. *Neural Computation*, 10(2):403–430, 1998.

[16] H. B. Barlow. Redundancy reduction revisited. *Network: Computation in Neural Systems*, 12:241–253, 2001.

[17] A. Hyvärinen. Fast and robust fixed-point algorithms for independent component analysis. *IEEE Trans. on Neural Networks*, 10(3):626–634, 1999.

[18] P. O. Hoyer. Modeling receptive fields with non-negative sparse coding. In E. De Schutter, editor, *Computational Neuroscience: Trends in Research 2003*. Elsevier, Amsterdam, 2003. In press.

[19] N. K. Logothetis and J. D. Schall. Neuronal correlates of subjective visual perception. *Science*, 245:761–763, 1989.

[20] S. Geman and D. Geman. Stochastic relaxation, gibbs distributions, and the bayesian restoration of images. *IEEE Transactions on Pattern Analysis and Machine Intelligence*, 6:721–741, 1984.

## Appendix: MCMC sampling of the non-negative ICA posterior

The posterior probability of $\mathbf{s}$, upon observing $\mathbf{x}$, is given by

$$p(\mathbf{s}|\mathbf{x}) = \frac{p(\mathbf{x}|\mathbf{s})p(\mathbf{s})}{p(\mathbf{x})} = C \exp\left(-\frac{1}{2\sigma^2}\|\mathbf{x} - \mathbf{As}\|^2\right) \prod_i \exp\left(-s_i\right). \tag{3}$$

Taking the (natural) logarithm yields

$$\log p(\mathbf{s}|\mathbf{x}) = \log C - \frac{1}{2\sigma^2}\left(\mathbf{x}^T\mathbf{x} - 2\mathbf{x}^T\mathbf{As} + \mathbf{s}^T\mathbf{A}^T\mathbf{As}\right) - \mathbf{u}^T\mathbf{s}, \tag{4}$$

where $\mathbf{u}$ is a vector of all ones. The crucial thing to note is that this function is quadratic in $\mathbf{s}$. Thus, the posterior distribution has the form of a gaussian, except that of course it is only defined for non-negative $\mathbf{s}$. Rejection sampling might look tempting, but unfortunately does not work well in high dimensions. Thus, we will instead opt for a Markov Chain Monte Carlo approach. Implementing Gibbs sampling [20] is quite straightforward. The posterior distribution of $s_k$, given $\mathbf{x}$ and all other hidden variables $s_j$, is a one-dimensional density that we will call *cut-gaussian*,

$$p\big(s_k|\mathbf{x}, s_{j\neq k}\big) \propto \begin{cases} 0 & \text{if } s_k < c_1 \\ \exp\left(-\frac{(s_k - \mu_{s_k})^2}{2\sigma_{s_k}^2}\right) & \text{if } c_1 \leq s_k \leq c_2 \\ 0 & \text{if } s_k > c_2 \end{cases} \tag{5}$$

In this case, we have the following parameter values:

$$\mu_{s_k} = \frac{\mathbf{a}_k^T(\mathbf{x} - \mathbf{As}') - \sigma^2}{\|\mathbf{a}_k\|^2}, \quad \sigma_{s_k} = \frac{\sigma}{\|\mathbf{a}_k\|}, \quad c_1 = 0, \text{ and } c_2 = \infty. \tag{6}$$

Here, $\mathbf{a}_k$ denotes the $k$:th column of $\mathbf{A}$, and $\mathbf{s}'$ denotes the current state vector but with $s_k$ set to zero. Sampling from such a one-dimensional distribution is relatively simple. Just as one can sample the corresponding (uncut) gaussian by taking uniformly distributed samples on the interval $(0,1)$ and passing them through the inverse of the gaussian cumulative distribution function, the same can be done for a cut-gaussian distribution by constraining the uniform sampling interval suitably.

Hence Gibbs sampling is feasible, but, as is well known, Gibbs sampling exhibits problems when there are significant correlations between the sampled variables. Thus we choose to use a sampling scheme based on a rotated co-ordinate system. The basic idea is to update the state vector not in the directions of the component axes, as in standard Gibbs sampling, but rather in the directions of the eigenvectors of $\mathbf{A}^T\mathbf{A}$. Thus we start by calculating these eigenvectors, and cycle through them one at a time. Denoting the current unit-length eigenvector to be updated $\mathbf{v}$ we have as a function of the step length $\alpha$,

$$\log p(\mathbf{s} + \alpha\mathbf{v}|\mathbf{x}) = \text{const} + \left(\frac{1}{\sigma^2}(\mathbf{x} - \mathbf{As})^T\mathbf{Av} - \mathbf{u}^T\mathbf{v}\right)\alpha - \frac{1}{2\sigma^2}\left(\mathbf{v}^T\mathbf{A}^T\mathbf{Av}\right)\alpha^2. \tag{7}$$

Again, note how this is a quadratic function of $\alpha$. Again, the non-negativity constraints on $\mathbf{s}$ require us to sample a cut-gaussian distribution. But this time there is an additional complication: When the basis is overcomplete, some of the eigenvectors will be associated with zero eigenvalues, and the logarithmic probability will be linear instead of quadratic. Thus, in such a case we must sample a *cut-exponential* distribution,

$$p(\alpha) \propto \begin{cases} 0 & \text{if } \alpha < c_1 \\ \exp\left(-\alpha/\mu_\alpha\right) & \text{if } c_1 \leq \alpha \leq c_2 \\ 0 & \text{if } \alpha > c_2 \end{cases} \tag{8}$$

Like in the cut-gaussian case, this can be done by uniformly sampling the corresponding interval and then applying the inverse of the exponential cumulative distribution function.

In summary: We start by calculating the eigensystem of the matrix $\mathbf{A}^T\mathbf{A}$, and set the state vector $\mathbf{s}$ to random non-negative values. Then we cycle through the eigenvectors indefinitely, sampling $\alpha$ from cut-gaussian or cut-exponential distributions depending on the eigenvalue corresponding to the current eigenvector $\mathbf{v}$, and updating the state vector $\mathbf{s}$ to $\mathbf{s} + \alpha\mathbf{v}$. MATLAB code performing and verifying this sampling is available at:

```
http://www.cis.hut.fi/phoyer/code/samplingpack.tar.gz
```